# Consistent Estimation of Functions of Data Missing Non-Monotonically and Not at Random

**Ilya Shpitser**
Department of Computer Science
Johns Hopkins University
ilyas@cs.jhu.edu

## Abstract

Missing records are a perennial problem in analysis of complex data of all types, when the target of inference is some function of the full data law. In simple cases, where data is missing at random or completely at random [15], well-known adjustments exist that result in consistent estimators of target quantities.

Assumptions underlying these estimators are generally not realistic in practical missing data problems. Unfortunately, consistent estimators in more complex cases where data is missing not at random, and where no ordering on variables induces monotonicity of missingness status are not known in general, with some notable exceptions [13, 18, 16].

In this paper, we propose a general class of consistent estimators for cases where data is missing not at random, and missingness status is non-monotonic. Our estimators, which are generalized inverse probability weighting estimators, make no assumptions on the underlying full data law, but instead place independence restrictions, and certain other fairly mild assumptions, on the distribution of missingness status conditional on the data.

The assumptions we place on the distribution of missingness status conditional on the data can be viewed as a version of a conditional Markov random field (MRF) corresponding to a chain graph. Assumptions embedded in our model permit identification from the observed data law, and admit a natural fitting procedure based on the pseudo likelihood approach of [2]. We illustrate our approach with a simple simulation study, and an analysis of risk of premature birth in women in Botswana exposed to highly active anti-retroviral therapy.

## 1   Introduction

Practical data sets generally have missing or corrupted entries. A classical missing data problem is to find a way to make valid inferences about the full data law. In other words, the goal is to exploit assumptions on the mechanism which is responsible for missingness or corruption of data records to transform the problem into another where *missingness or corruption were not present at all*.

In simple cases, where missingness status is assumed to be missing completely at random (determined by an independent coin flip), or at random (determined by a coin flip independent conditional on observed data records), adjustments exist which result in consistent estimators of many functions of the full data law. Unfortunately, these cases are difficult to justify in practice. Often, data records are missing intermittently and in complex patterns that do not conform to above assumptions. For instance, in longitudinal observational studies in medicine, patients may elect to not show up at a particular time point, for reasons having to do with their (by definition missing) health status *at that time point*, and then later return for followup.

In this situation, missingness is not determined by a coin flip independent of missing data conditional on the observed data (data is missing not at random), and missingness status of a patient is not monotonic under any natural ordering. In this setting, deriving consistent estimators of even simple functions of the full data law is a challenging problem [13, 18, 16].

In this paper we propose a new class of consistent generalized inverse probability weighting (IPW) estimators for settings where data is missing non-monotonically and not at random. Like other IPW estimators, ours makes no modeling assumptions on the full data law, and only models the joint missingness status of all variables, conditional on those variables. This model can be viewed as a conditional Markov random field (MRF) with independence assumptions akin to those made in factors of a chain graph model [6]. The assumptions encoded in our model permit identification of the full data law, and allow estimation based on the pseudo likelihood approach of [2].

Our paper is organized as follows. We discuss relevant preliminaries on graphical models in Section 2. We fix additional notation and discuss some prior work on missing data in Section 3. We introduce our missingness model, and identification results based on it in Section 4, and discuss estimation in Section 5. We illustrate the use of our model with a simple simulation study in Section 6, and give a data analysis application in Section 7. Finally, we illustrate the difference between our model and a seemingly similar non-identified model via a parameter counting argument in Section 8, and give our conclusions in Section 9.

## 2    Chain Graph Models

We briefly review statistical chain graph models. A simple mixed graph is a graph where every vertex pair is connected by at most one edge, and there are two types of possible edges: directed and undirected. Chain graphs are mixed graphs with the property that for every edge cycle in the graph, there is no way to assign an orientation to undirected edges in any cycle to form a directed cycle [6].

For a graph $\mathcal{G}$ with a vertex set $\mathbf{V}$, and any subset $\mathbf{A} \subseteq \mathbf{V}$, define the *induced subgraph* $\mathcal{G}_{\mathbf{A}}$ to be a graph with the vertex set $\mathbf{A}$ and all edges in $\mathcal{G}$ between elements in $\mathbf{A}$. Given a graph $\mathcal{G}$, define the *augmented* or *moral* graph $\mathcal{G}^a$ to be an undirected graph obtained from adding a new undirected edge between any unconnected vertices $W_1, W_2$ if a path of the form $W_1 \rightarrow \circ - \circ \ldots \circ -\circ \leftarrow W_2$ exists in $\mathcal{G}$ (note the path may only contain a single intermediate vertex), and then replacing all directed edges in $\mathcal{G}$ by undirected edges.

A *clique* in an undirected graph is a set of vertices where any pair of vertices are neighbors. A maximal clique is a clique such that no superset of it forms a clique. Given an undirected graph $\mathcal{G}$, denote the set of maximal cliques in $\mathcal{G}$ by $\mathcal{C}(\mathcal{G})$. A *block* in a simple mixed graph $\mathcal{G}$ is any connected component formed by undirected edges in a graph obtained from $\mathcal{G}$ dropping all directed edges. Given a simple mixed graph $\mathcal{G}$, denote the set of blocks in $\mathcal{G}$ by $\mathcal{B}(\mathcal{G})$.

A chain graph model is defined by the following factorization

$$p(\mathbf{V}) = \prod_{\mathbf{B} \in \mathcal{B}(\mathcal{G})} p(\mathbf{B} \mid \mathrm{pa}_{\mathcal{G}}(\mathbf{B})), \tag{1}$$

where for each $\mathbf{B}$,

$$p(\mathbf{B} \mid \mathrm{pa}_{\mathcal{G}}(\mathbf{B})) = \frac{1}{\mathbf{Z}(\mathrm{pa}_{\mathcal{G}}(\mathbf{B}))} \prod_{\mathbf{C} \in \mathcal{C}((\mathcal{G}_{\mathbf{B} \cup \mathrm{pa}_{\mathcal{G}}(\mathbf{B})})^a)} \phi_{\mathbf{C}}(\mathbf{C}), \tag{2}$$

and $\phi_{\mathcal{C}}(\mathbf{C})$ are called *potential functions* and map value configurations of variables in $\mathbf{C}$ to real numbers, which are meant to denote an "affinity" of the model towards that particular value configuration. The chain graph factorization implies Markov properties, described in detail in [6].

## 3    Preliminaries, and Prior Work on Missing Data

We will consider data sets on random variables $\mathbf{L} \equiv L_1, \ldots L_k$, drawn from a full data law $p(\mathbf{L})$. Associated with each random variable $L_i$ is a binary missingness indicator $R_i$, where $L_i$ is observed if and only if $R_i = 1$. Define a vector $(\mathbf{l}^j, \mathbf{r}^j) \equiv (l_1^j, \ldots l_k^j, r_1^j, \ldots r_k^j)$ to be the $j$th realization of

$p(\mathbf{L}, \mathbf{R})$. Define $(\mathbf{l}^*)^j \equiv \{l_i^j \mid r^j = 1\} \subseteq \mathbf{l}^j$. In missing data settings, for every $j$, we only get to observe the vector of values $((\mathbf{l}^*)^j, \mathbf{r}^j)$, and we wish to make inferences using the true realizations $(\mathbf{l}^j, \mathbf{r}^j)$ from the underlying law. Doing this entails building a bridge between the observed and the underlying realizations, and this bridge is provided by assumptions made on $p(\mathbf{L}, \mathbf{R})$.

If we can assume that for any $i$, $p(R_i \mid \mathbf{L}) = p(R_i)$, in other words, every missing value is determined by an independent biased coin flip, then data is said to be *missing completely at random* (MCAR) [15]. In this simple setting, it is known that any estimator for complete data remains consistent if applied to just the complete cases. A more complex assumption, known as *missing at random* (MAR) [15], states that for every $i$, $p(R_i \mid \mathbf{L}) = p(R_i \mid \mathbf{L}^*)$. In other words, every missing value is determined by a biased coin flip that is independent of missing data values conditional on the observed data values. In this setting, a variety of adjustments lead to consistent estimators.

The most interesting patterns of missingness, and the most relevant in practice, are those that do not obey either of the above two assumptions, in which case data is said to be *missing not at random* (MNAR). Conventional wisdom in MNAR settings is that without strong parametric modeling assumptions, many functions of the full data law are not identified from the observed data law. Nevertheless, a series of recent papers [8, 7, 17], which represented missing data mechanisms as graphical models, and exploited techniques developed in causal inference, have shown that the full data law may be non-parametrically identified under MNAR.

In this approach, the full data law is assumed to factorize with respect to a directed acyclic graph (DAG) [11]. Assumptions implied by this factorization are then used to derive functions of $p(\mathbf{L})$ in terms of $p(\mathbf{R}, \mathbf{L}^*)$. We illustrate the approach using Fig. 1 (a),(b) and (c). Here nodes in green are assumed to be completely observed. In Fig. 1 (a), the Markov factorization is $p(R_2, L_1, L_2) = p(R_2 \mid L_1)p(L_2 \mid L_1)p(L_1)$. It is easy to verify using d-separation [11] in this DAG that $p(R_2 \mid L_1, L_2) = p(R_2 \mid L_1)$. Since $L_1$ is always observed, this setting is MAR, and we get the following $p(L_1, L_2) = p(L_2|L_1)p(L_1) = p(L_2|L_1, R_2 = 1)p(L_1) = p(R_2 = 1, \mathbf{L}^*)/p(R_2 = 1|L_1)$, where the last expression is a functional of $p(\mathbf{R}, \mathbf{L}^*)$, and so the full data law $p(\mathbf{L})$ is non-parametrically identified from the observed data law $p(\mathbf{R}, \mathbf{L}^*)$.

The ratio form of the identifying functional suggests the following simple IPW estimator for $E[L_2]$, known as the Horvitz-Thompson estimator [4]. We estimate $p(R_2 \mid L_1)$ either directly if $L_1$ is discrete and low dimensional, or using maximum likelihood fitting of a model for $p(R_2 \mid L_1; \boldsymbol{\beta})$, for instance a logistic regression model. We then average observed values of $L_2$, but compensate for the fact that observed and missing values of $L_2$ *systematically differ* using the inverse of the fitted probability of the case being observed, conditional on $L_1$, or $\hat{E}[L_2] = N^{-1} \sum_{n:r^n=1} L_2^n/p(R_2 = 1 \mid l_1^n; \hat{\boldsymbol{\beta}})$. Under our missingness model, this estimator is clearly unbiased. Under a number of additional fairly mild conditions, this estimator is also consistent.

A more complicated graph, shown in Fig. 1 (b), implies the following factorization

$$p(L_1, L_2, R_1, R_2) = p(R_1 \mid L_2)p(R_2 \mid L_1)p(L_1 \mid L_2)p(L_2). \tag{3}$$

Using d-separation in this DAG, we see that in cases where any values are missing, neither MCAR nor MAR assumptions hold under this model. Thus, in this example, data is MNAR. However, the conditional independence constraints implied by the factorization (3) imply the following

$$p(L_1, L_2) = \frac{p(R_1 = 1, R_2 = 1, \mathbf{L}^*)}{p(R_1 = 1 \mid L_2^*, R_2 = 1) \cdot p(R_2 = 1 \mid L_1^*, R_1 = 1)}.$$

As before, all terms on the right hand side are functions of $p(\mathbf{R}, \mathbf{L}^*)$, and so $p(\mathbf{L})$ is non-parametrically identified from $p(\mathbf{R}, \mathbf{L}^*)$. This example was discussed in [8].

The form of the identifying functional suggests a simple generalization of the IPW estimator from the previous example for $E[L_2]$. As before, we fit models $p(R_1 \mid L_2^*; \boldsymbol{\beta}_1)$ and $p(R_2 \mid L_1^*; \boldsymbol{\beta}_2)$ by MLE. We take the empirical average of the observed values of $L_2$, but reweigh them by the inverses of *both* of the estimated probabilities, using complete cases only:

$$\frac{1}{N} \sum_{n:r_1^n = r_2^n = 1} l_2^n \cdot \frac{1}{p(r_1 = 1 \mid l_2^n; \hat{\boldsymbol{\beta}}_1) \cdot p(r_1 = 1 \mid l_1^n; \hat{\boldsymbol{\beta}}_2)}.$$

This estimator is also consistent, with the proof a simple generalization of that for Horvitz-Thompson. More generally, it has been shown in [8] that in DAGs where no $R$ variable has a

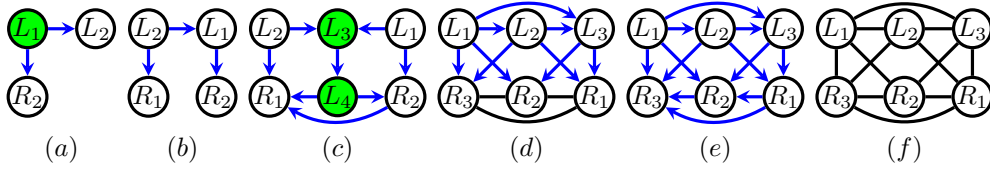

Figure 1: (a) A graphical model for MAR data. (b),(c) Graphical model for MNAR data where identification of the full data law is possible. (d) The no self-censoring model for $k = 3$. (e) A missingness model seemingly similar to (d), where the full data law is not identified. (f) An undirected graph representing an independence model Markov equivalent to the independence model represented by a chain graph in (d).

child, and the edge $L_i \to R_i$ does not exist for any $i$, we get:

$$p(\mathbf{L}) = \frac{p(\mathbf{L}^*, \mathbf{R} = \mathbf{1})}{\prod_{R_i} p(R_i \mid \mathrm{pa}_{\mathcal{G}}(R_i), \mathbf{R}_{\{i|L_i \in \mathrm{pa}_{\mathcal{G}}(R_i)\}} = \mathbf{1})}.$$

This identifying functional implies consistent IPW estimators can be derived that are similar to estimators in the above examples.

The difficulty with this result is that it assumes missingness indicators are disconnected. This assumption means we cannot model *persistent dropout* or *loss to followup* (where $R_i = 0$ at one time point implies $R_i = 0$ at all following time points), or complex patterns of non-monotone missing data (where data is missing intermittently, but missingness also exhibits complex dependence structure). This kind of dependence is represented by connecting $R$ variables in the graph. Unfortunately, this often leads to non-identification – the functional of the full data law not being a function of the observed data law. For instance, if we add an edge $R_1 \to R_2$ to Fig. 1 (b), it is known that $p(L_1, L_2)$ is not identified from $p(\mathbf{R}, \mathbf{L}^*)$. Intuition for this is presented in Section 8.

A classical approach to missingness with connected $R$ variables assumes sequential ignorability, and monotone missingness (where there exists an ordering on variables such that every unit that's missing earlier in the ordering remains missing later in the ordering) [12]. However, this approach does not easily generalize to data missing in non-monotone patterns and not at random.

Nevertheless, if a sufficient number of edges are missing in the graph, identification sometimes is possible even if $R$ variables are dependent, and monotonicity and MAR do not hold. In particular, techniques from causal inference have been using to derive complex identification results in this setting [7, 17]. For instance, it has been shown that in Fig. 1 (c), $p(L_1, L_2, L_3, L_4) = \frac{p(\mathbf{L}^*, \mathbf{R}=\mathbf{1})}{\tilde{p}_1 \cdot \tilde{p}_2}$, where $\tilde{p}_1 = q_{L_4}(R_1 = 1 \mid L_2, R_2 = 1)$, $\tilde{p}_2 = \frac{q_{L_4}(L_1|R_2=1,R_1=1)q_{L_4}(R_2=1)}{\sum_{R_2} q_{L_4}(L_1|R_2,R_1=1)q_{L_4}(R_2)}$ and $q_{L_4}(R_1, R_2, L_1, L_2, L_3) = p(L_1, L_2, R_1, R_2 \mid L_3, L_4)p(L_3)$. See [17] for details. Unfortunately, it is often difficult to give a practical missing data setting which exhibits the particular pattern of missing edges that permits identification. In addition, a full characterization of identifiability of functionals of the full data law under MNAR is an open problem. In the next sections, we generalize the graphical model approach to missing data from DAGs to a particular type of chain graph. Our model is able to encode fairly general settings where data is missing non-monotonically and not at random, while also permitting identification of the full data law under fairly mild assumptions.

## 4 The No Self-Censoring Missingness Model

Having given the necessary preliminaries, we are ready to define our missingness model for data missing non-monotonically and not at random. Our desiderata for such a model are as follows. First, in order for our model to be useful in as wide a variety of missing data settings as possible, we want to avoid imposing any assumptions on the underlying full data law. Second, since we wish to consider arbitrary non-monotonic missingness patterns, we want to allow arbitrary relationships between missingness indicators. Finally, since we wish to allow data to be missing not at random, we want to allow as much dependence of missingness indicators on the underlying data, even if missing, as possible.

However, a completely unrestricted relationship between underlying variables and missingness indicators can easily lead to non-identification. For instance in any graph where the edge $L_i \rightarrow R_i$ exists, the marginal distribution $p(L_i)$ is not in general a function of the observed data law. Thus, we do not allow variables to drive their own missingness status, and thus edges of the form $L_i \rightarrow R_i$. However, we allow a variable to influence its own missingness status *indirectly*.

Surprisingly, the restrictions given so far essentially characterize independences defining our proposed model. Consider the following chain graph on vertices $L_1, \ldots L_k, R_1, \ldots R_k$. The vertices $L_1, \ldots, L_k$ form a complete DAG, meaning that the full data law $p(L_1, \ldots, L_k)$ has no restrictions. The vertices $R_1, \ldots R_k$ form a k-clique, meaning arbitrary dependence structure between $R$ variables is allowed. In addition, for every $i$, $\mathrm{pa}_{\mathcal{G}}(R_i) \equiv \mathbf{L} \setminus \{L_i\}$, which restricts a variable $L_i$ from directly causing its own missingness status $R_i$. The resulting graph is always a chain graph. An example (for $k = 3$) is shown in Fig. 1 (c). The factorizations (1) and (2) for chain graphs of this form imply a particular set of independence constraints.

**Lemma 1** *Let $\mathcal{G}$ be a chain graph with vertex set $\mathbf{R} \cup \mathbf{L}$, where $\mathcal{B}(\mathcal{G}) = \{\mathbf{R}, \{L_1\}, \ldots \{L_k\}\}$, and for every $i$, $\mathrm{pa}_{\mathcal{G}}(L_i) = \{L_1, \ldots L_{i-1}\}$, $\mathrm{pa}_{\mathcal{G}}(R_i) = \mathbf{L} \setminus \{L_i\}$. Then for every $i$, and every $p(\mathbf{L}, \mathbf{R})$ that factorizes according to $\mathcal{G}$, the only conditional independences implied by this factorization on $p(\mathbf{L}, \mathbf{R})$ are $(\forall i) \, (R_i \perp\!\!\!\perp L_i \mid \mathbf{R} \setminus \{R_i\}, \mathbf{L} \setminus \{L_i\})$.* [1]

*Proof:* This follows by the global Markov property results for chain graphs, found in [6]. □

This set of independences in $p(\mathbf{R}, \mathbf{L})$ can be represented not only by a chain graph, but also by an undirected graph where every pair vertices except $R_i$ and $L_i$ (for every $i$) are connected. Such a graph, interpreted as a Markov random field, would imply the same set of conditional independences as those in Lemma 1. An example of such a graph for $k = 3$ is shown in Fig. 1 (f). The reason we emphasize viewing the model using chain graphs is because the only independence restrictions we place are on the conditional distribution $p(\mathbf{R} \mid \mathbf{L})$; these restrictions resemble those found in factors of (1), and not in classical conditional Markov random fields, where every variable in $\mathbf{R}$ would depend on every variable in $\mathbf{L}$. We call the missingness model with this independence structure the *no self-censoring model*, due to the fact that no variable $L_i$ is allowed to directly censor itself via setting $R_i$ to 0. We now show that under relatively benign assumptions, we can identify the full data law $p(\mathbf{L})$ in this model.

**Lemma 2** *If $p(\mathbf{R} = \mathbf{1} \mid \mathbf{L})$ is identified from the observed data distribution $p(\mathbf{L}^*, \mathbf{R} = \mathbf{1})$, then $p(\mathbf{L})$ is identified from $p(\mathbf{L}^*, \mathbf{R} = \mathbf{1})$ via $p(\mathbf{L}^*, \mathbf{R} = \mathbf{1})/p(\mathbf{R} = \mathbf{1} \mid \mathbf{L})$.*

*Proof:* Trivially follows by the chain rule of probability, and the fact that $\mathbf{L} = \mathbf{L}^*$ if $\mathbf{R} = \mathbf{1}$. □

To obtain identification, we use a form of the *log conditional pseudo-likelihood (LCPL) function*, first considered (in joint form) in [2]. Define, for any parameterization $p(\mathbf{R} \mid \mathbf{L}; \boldsymbol{\alpha})$, where $|\mathbf{R}| = k$,

$$\log \mathcal{PL}(\boldsymbol{\alpha}) = \sum_{i=1}^{k} \sum_{j:\mathbf{L}^j \setminus \{L_i^j\} \subseteq (\mathbf{L}^*)^j} \log p(R_i = r_i^j \mid \mathbf{R}^j \setminus \{R_i^j\} = \mathbf{r}^j, \mathbf{L}^j; \boldsymbol{\alpha}).$$

In subsequent discussion we will assume that if $p_1(\mathbf{R} \mid \mathbf{L}; \boldsymbol{\alpha}_0) \neq p_2(\mathbf{R} \mid \mathbf{L}; \boldsymbol{\alpha})$ then $\boldsymbol{\alpha}_0 \neq \boldsymbol{\alpha}$.

**Lemma 3** *Under the no self-censoring model, in the limit of infinite data sampled from $p(\mathbf{R}, \mathbf{L})$, where only $\mathbf{L}^*, \mathbf{R}$ is observed, $\log \mathcal{PL}(\boldsymbol{\alpha})$ is maximized at the true parameter values $\boldsymbol{\alpha}_0$.*

*Proof:* The proof follows that for the standard pseudo-likelihood in [9]. The difference between the LCPL functions evaluated at $\boldsymbol{\alpha}_0$ and $\boldsymbol{\alpha}$ can be expressed as a sum of conditional relative entropies, which is always non-negative. The fact that every term in the LCPL function is a function of the observed data follows by Lemma 1. □

We will restrict attention to function classes which satisfy standard assumptions needed to derive consistent estimators [10], namely compactness of the parameter space, dominance, and (twice) differentiability with respect to $\boldsymbol{\alpha}$, which implies continuity.

**Corollary 1** *Under the no self-censoring model of missingness, and assumptions above, the estimator of $\boldsymbol{\alpha}$ maximizing the LCPL function is weakly consistent.*

*Proof:* Follows by Lemma 3, and the argument in [9] via equation (9), Lemma 1 and Theorem 1. □

## 5   Estimation

Since all variables in $\mathbf{R}$ are binary, and our model for $p(\mathbf{R} \mid \mathbf{L})$ is a type of conditional MRF, a log-linear parameterization is natural. We thus adapt the following class of parameterizations:

$$p(\mathbf{R} = \mathbf{r} \mid \mathbf{L} = \mathbf{l}) = \frac{1}{Z(\mathbf{l})} \exp \left\{ \sum_{\mathbf{R}^\dagger \subseteq \mathcal{P}(\mathbf{R}) \setminus \{\emptyset\}} \mathbf{r}_{\mathbf{R}^\dagger} \cdot f_{\mathbf{R}^\dagger}(\mathbf{l}_{\mathbf{L} \setminus \mathbf{L}^\dagger}; \alpha_{\mathbf{R}^\dagger}) \right\} \tag{4}$$

where $\mathbf{L}^\dagger \equiv \{L_i \mid R_i \in \mathbf{R}^\dagger\}$, $\mathcal{P}(\mathbf{R})$ is the powerset of $\mathbf{R}$, and for every $\mathbf{R}^\dagger$, $f_{\mathbf{R}^\dagger}$ is a function parameterized by $\alpha_{\mathbf{R}^\dagger}$, mapping values of $\mathbf{L} \setminus \mathbf{L}^\dagger$ to an $|\mathbf{R}^\dagger|$-way interaction. Let $\boldsymbol{\alpha} \equiv \{\alpha_{\mathbf{R}^\dagger} \mid \mathbf{R}^\dagger \subseteq \mathcal{P}(\mathbf{R}) \setminus \{\emptyset\}\}$. We now show our class of parameterizations gives the right independence structure.

**Lemma 4** *For an arbitrary $p(\mathbf{L})$, and a conditional distribution $p(\mathbf{R} \mid \mathbf{L})$ parameterized as in (4), the set of independences in Lemma 1 hold in the joint distribution $p(\mathbf{L}, \mathbf{R}) = p(\mathbf{R} \mid \mathbf{L})p(\mathbf{L})$.*

*Proof:* For any $R_i \in \mathbf{R}$, and values $\mathbf{r}, \mathbf{l}$, such that $\mathbf{r}_{R_i} = 1$,

$$p(\mathbf{r}_{R_i} \mid \mathbf{r}_{\mathbf{R} \setminus \{R_i\}}, \mathbf{l}_{\mathbf{L}}) = \frac{\exp \left\{ \sum_{R_i \in \mathbf{R}^\dagger \subseteq \mathcal{P}(\mathbf{R}) \setminus \{\emptyset\}} \mathbf{r}_{\mathbf{R}^\dagger} \cdot f_{\mathbf{R}^\dagger}(\mathbf{l}_{\mathbf{L} \setminus \mathbf{L}^\dagger}; \alpha_{\mathbf{R}^\dagger}) \right\}}{1 + \exp \left\{ \sum_{R_i \in \mathbf{R}^\dagger \subseteq \mathcal{P}(\mathbf{R}) \setminus \{\emptyset\}} \mathbf{r}_{\mathbf{R}^\dagger} \cdot f_{\mathbf{R}^\dagger}(\mathbf{l}_{\mathbf{L} \setminus \mathbf{L}^\dagger}; \alpha_{\mathbf{R}^\dagger}) \right\}}.$$

By definition of $f_{\mathbf{R}^\dagger}$, this functional is not a function of $L_i$, which gives our result.             □

As expected with a log-linear conditional MRF, the distribution $p(R_i \mid \mathbf{R} \setminus \{R_i\}, \mathbf{L})$ resembles the logistic regression model. Under twice differentiability of $f_{\mathbf{R}^\dagger}$, first and second derivatives of the LCPL function have a straightforward derivation, which we omit in the interests of space. Just as with the logistic model, the estimating equations cannot be solved in closed form, but iterative algorithms are straightforward to construct. For sufficiently simple $f_{\mathbf{R}^\dagger}$, the Newton-Raphson algorithm may be employed. Note that every conditional model for $R_i$ is fit only using rows where $\mathbf{L} \setminus \{L_i\}$ are observed. Thus, the fitting procedure fails in datasets with few enough samples that for some $R_i$, no such rows exist. We leave extensions of our model that deal with this issue to future work.

Finally, we use our fitted model $p(\mathbf{R} \mid \mathbf{L}; \hat{\boldsymbol{\alpha}})$, as a joint IPW for estimating functions of $p(\mathbf{L})$. For instance, if $L_1, \ldots L_{k-1}$ represents intermediate outcomes, and $L_k$ the final outcome of a longitudinal study with intermittent MNAR dropout represented by our model, and we are interested in the expected final outcome, $E[L_k]$, we would extend IPW estimators discussed in Section 3 as follows: $\hat{E}[L_k] = N^{-1} \sum_{n: \mathbf{r}^n = \mathbf{1}} l_k^n / p(\mathbf{R} = \mathbf{1} \mid \mathbf{l}^n; \hat{\boldsymbol{\alpha}})$. Estimation of more complex functionals of $p(\mathbf{L})$ proceeds similarly, though it may employ marginal structural models if $\mathbf{L}$ is high-dimensional. Consistency of these estimators follows, under the usual assumptions, by standard arguments for IPW estimators, and Corollary 1.

## 6   A Simple Simulation Study

To verify our results, we implemented our estimator for a simple model in the class of parameterizations (4) that satisfy the assumptions needed for deriving the true parameter by maximizing the LCPL function. Fig. 2 shows our results. For the purposes of illustration, we chose the model in Fig. 1 (d) with functions $f_{\mathbf{R}^\dagger}$ defined as follows. For every edge $(L_i, R_j)$ in the graph, define a parameter $w_{ij}$, and a parameter $w_\emptyset$. Define every function $f_{\mathbf{R}^\dagger}$ to be of the form $\sum_{i: L_i \in \mathbf{L} \setminus \mathbf{L}^\dagger, j: R_j \in \mathbf{R}^\dagger} w_{ij} L_i(1)$. The values of $L_1, L_2, L_3$ were drawn from a multivariate normal distribution with parameters $\mu = (1, 1, 1), \Sigma = I + 1$. We generated a series of datasets with sample size 100 to 1000, and compared differences between the true means $E[L_i(1)]$ and the unadjusted (complete case) MLE estimate of $E[L_i(1)]$ (blue), and IPW adjusted estimate of $E[L_i(1)]$ (red), for $i = 1, 2, 3$. The true difference is, of course, 0. Confidence intervals at the 95% level were computing using case resampling bootstrap (50 iterations). The confidence intervals generally overlapped

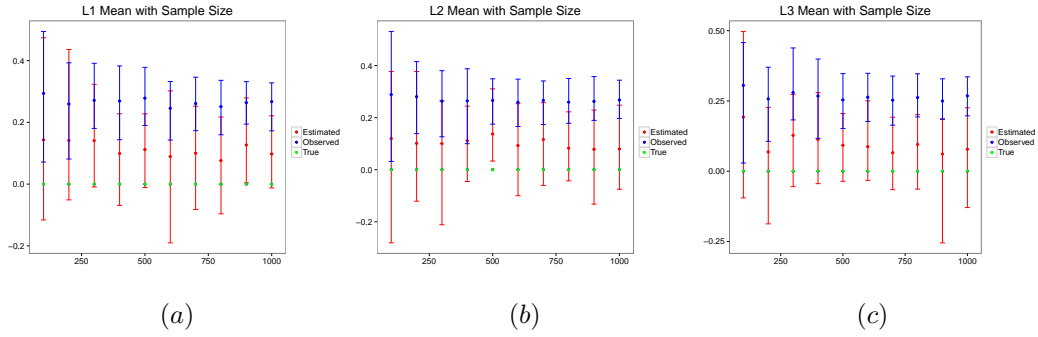

$$(a) \qquad\qquad (b) \qquad\qquad (c)$$

Figure 2: (a),(b),(c) Results of estimating $E[L_1(1)]$, $E[L_2(1)]$ and $E[L_3(1)]$, respectively, from a model in Fig. 1 (d). Y axis is parameter value, and $X$ axis is sample size. Confidence intervals are reported using case resampling bootstrap at 95% level. Confidence interval size does not necessarily shrink with sample size – a known issue with IPW estimators.

0, while complete case analysis did not. We noted that confidence intervals did not always shrink with increased sample size – a known difficulty with IPW estimators.

Aside from the usual difficulties with IPW estimators, which are known to suffer from high variance, our estimator only reweighs observed cases, which may in general be a small fraction of the overall dataset as $k$ grows (in our simulations only 50-60% of cases were complete). Furthermore, estimating weights by maximizing pseudo-likelihood is known to be less efficient than by maximizing likelihood, since all variability of variables in the conditioning sets is ignored.

## 7   Analysis Application

To illustrate the performance of our model in a practical setting where data is missing not at random, we report an analysis of a survey dataset for HIV-infected women in Botswana, also analyzed in [18]. The goal is to estimate an association between maternal exposure to highly active anti-retroviral therapy (HAART) during pregnancy and a premature birth outcome among HIV-infected women in Botswana. The overall data consisted of 33148 obstetrical records from 6 cites in Botswana. Here we restricted to a subset of HIV positive women ($n = 9711$). We considered four features: the outcome (preterm delivery), with $6.7\%$ values missing, and two risk factors – whether the CD4 count (a measure of immune system health) was lower than 200 cells per $\mu L$ (53.1% missing), and whether HAART was continued from before pregnancy (69.0% missing). We also included hypertension – a common comorbidity of HIV (6.5% missing). In this dataset missing at random is not a reasonable assumption, and what's more missingness patterns are not monotonic.

We used a no-self censoring model with $f_{\mathbf{R}^\dagger}(.)$ of the same form as in section 6. The results are shown in Fig. 3, which contain the complete case analysis (CC), the no self-censoring model (NSCM), and a version of the discrete choice model in [18] (DCM). We reported the odds ratios (ORs) with a 95% confidence interval, obtained by bootstrap. Note that CC and DCM confidence intervals for the OR overlap 1, indicating a weak or non-existent effect. The confidence interval for the NSCM indicates a somewhat non-intuitive inverse relationship for low CD4 count and premature birth, which we believe may be due to assumptions of the NSCM not being met with a limited set of four variables we considered. In fact, the dataset was sufficiently noisy that an expected positive relationship was not found by any method.

## 8   Parameter Counting

Parameter counting may be used to give an intuition for why $p(\mathbf{L})$ is identified under the no self-censoring model, but not under a very similar missingness model where undirected edges between $R$ variables are replaced by directed edges under some ordering (see Fig. 1 (d) and

|      | Low CD4 Count          | Cont HAART             |
|------|------------------------|------------------------|
| CC   | 0.782 (0.531, 1.135)   | 1.142 (0.810, 1.620)   |
| NSCM | 0.651 (0.414, 0.889)   | 1.032 (0.670, 1.394)   |
| DCM  | 1.020 (0.742, 1.397)   | 1.158 (0.869, 1.560)   |

Figure 3: Analyses of the HIV pregnancy Botswana dataset. CC: complete case analysis, NSCM: the no self-censoring model with a linear parameterization, DCM: a member of the discrete choice model family described in [18].

(e) for an example for $k = 3$.) Assume $|\mathbf{L}| = k$, where $L$ variables are discrete with $d$ levels. Then the observed data law may be parameterized by $2^k - 1$ parameters for $p(\mathbf{R})$, and by $d^{k-|\mathbf{R}^\dagger|-1}$ parameters for each $p(\mathbf{L}^* \mid \mathbf{R}^\dagger = \mathbf{1}, \mathbf{R} \setminus \mathbf{R}^\dagger = \mathbf{0})$, where $\mathbf{R}^\dagger \neq \emptyset$, for a total of $2^k - 1 + \sum_{\mathbf{R}^\dagger \subseteq \mathcal{P}(\mathbf{R}) \setminus \{\emptyset\}} \binom{k}{|\mathbf{R}^\dagger|}(d^{|\mathbf{R}^\dagger|} - 1) = (d+1)^k - 1$. The no-censoring model needs $d^k - 1$ parameters for $p(\mathbf{L})$, and $\sum_{\mathbf{R}^\dagger \in \mathcal{P}(\mathbf{R}) \setminus \{\emptyset\}} \binom{k}{|\mathbf{R}^\dagger|} d^{k-|\mathbf{R}^\dagger|}$ for $p(\mathbf{R} \mid \mathbf{L})$, yielding a total of $d^k - 1 + (d+1)^k - d^k = (d+1)^k - 1$, which means the model is just-identified, and imposes no restrictions on the observed data law under our assumptions on $\mathbf{L}$. However, the DAG model needs $d^k - 1$ parameters for $p(\mathbf{L})$, and $\sum_{i=1}^{k}(d^{k-1} \cdot 2^{i-1})$ for $p(\mathbf{R} \mid \mathbf{L})$, for a total of $d^k - 1 + d^{k-1} \cdot (2^k - 1)$. The following Lemma implies the DAG version of the no self-censoring model is not identified.

**Lemma 5** $d^{k-1} \cdot (2^k - 1) > (d+1)^k - d^k$ for $k \geq 2, d \geq 2$.

*Proof:* For $k = 2$, we have $3d > 2d + 1$, which holds for any $d > 1$. If our result holds for $k$, then $2^k > (d+1)^k/d^{k-1} - d + 1$. Then the inequality holds for $k + 1$, since $2 > (d+1)/d$ for $d > 1$. $\square$

Just identification under the independence structure given in Lemma 1 was used in [16] (independently of this paper) to derive a parameterization of the model that uses the observed data law. This paper, by contrast, only models the missingness process represented by $p(\mathbf{R} \mid \mathbf{L})$, and does not model the observed data law $p(\mathbf{L}^*)$ at all.

## 9 Conclusions

In this paper, we have presented a graphical missingness model based on chain graphs for data missing non-monotonically and not at random. Specifically, our model places no restrictions on the underlying full data law, and on the dependence structure of missingness indicators, and allows a high degree of interdependence between the underlying unobserved variables and missingness indicators. Nevertheless, under our model, and fairly mild assumptions, the full data law is identified. Our estimator is an inverse probability weighting estimator with the weights being joint probabilities of the data being observed, conditional on all variables. The weights are fitted by maximizing the log conditional pseudo likelihood function, first derived in joint form in [2].

We view our work as an alternative to existing and newly developed methods for MNAR data [13, 18, 16], and an attempt to bridge the gap between the existing rich missing data literature on identification and estimation strategies for MAR data (see [14] for further references), and newer work which gave an increasingly sophisticated set of identification conditions for MNAR data using missingness graphs [8, 7, 17]. The drawbacks of existing MAR methods is that most missingness patterns of practical interest are not MAR, the drawbacks of the missingness graph literature is that it has not yet considered estimation, and used assumptions on missingness that, while MNAR, are difficult to justify in practice (for example Fig. 1 (c) implies a complicated identifying functional under MNAR, but places a marginal independence restriction ($L_1 \perp\!\!\!\perp L_2$) on the full data law).

Our work remedies both of these shortcomings. On the one hand, we assume a very general, and thus easier to justify in practice, missingness model for MNAR data. On the other, we don't just consider an identification problem for our model, but give a class of IPW estimators for functions of the observed data law. Addressing statistical and computational challenges posed by our class of estimators, and making them practical for analysis of high dimensional MNAR data is our next step.

## Footnotes

[1] $\mathbf{A} \perp\!\!\!\perp \mathbf{B} \mid \mathbf{C}$ is notation found in [3], meaning $\mathbf{A}$ is independent of $\mathbf{B}$ given $\mathbf{C}$.

# References

[1] Heejung Bang and James M. Robins. Doubly robust estimation in missing data and causal inference models. *Biometrics*, 61:962–972, 2005.

[2] Julian Besag. Statistical analysis of lattice data. *The Statistician*, 24(3):179–195, 1975.

[3] A. Philip Dawid. Conditional independence in statistical theory. *Journal of the Royal Statistical Society*, 41:1–31, 1979.

[4] D. G. Horvitz and D. J. Thompson. A generalization of sampling without replacement from a finite universe. *Journal of the American Statistical Association*, 47:663–685, 1952.

[5] J. Lafferty, A. McCallum, and F. Pereira. Conditional random fields: Probabilistic models for segmenting and labeling sequence data. In *Proceedings of the Eighteenth International Conference on Machine Learning (ICML-01)*, pages 282 – 289. Morgan Kaufmann, 2001.

[6] Steffan L. Lauritzen. *Graphical Models*. Oxford, U.K.: Clarendon, 1996.

[7] Karthika Mohan and Judea Pearl. Graphical models for recovering probabilistic and causal queries from missing data. In Z. Ghahramani, M. Welling, C. Cortes, N.D. Lawrence, and K.Q. Weinberger, editors, *Advances in Neural Information Processing Systems 27*, pages 1520–1528. Curran Associates, Inc., 2014.

[8] Karthika Mohan, Judea Pearl, and Jin Tian. Graphical models for inference with missing data. In C.J.C. Burges, L. Bottou, M. Welling, Z. Ghahramani, and K.Q. Weinberger, editors, *Advances in Neural Information Processing Systems 26*, pages 1277–1285. Curran Associates, Inc., 2013.

[9] A. Mozeika, O. Dikmen, and J. Piili. Consistent inference of a general model using the pseudolikelihood method. *Physical Review E: Statistical, Nonlinear, and Soft Matter Physics.*, 90, 2014.

[10] Whitney Newey and Daniel McFadden. Chapter 35: Large sample estimation and hypothesis testing. In *Handbook of Econometrics, Vol.4*, pages 2111–2245. Elsevier Science, 1994.

[11] Judea Pearl. *Probabilistic Reasoning in Intelligent Systems*. Morgan and Kaufmann, San Mateo, 1988.

[12] James M. Robins. A new approach to causal inference in mortality studies with sustained exposure periods – application to control of the healthy worker survivor effect. *Mathematical Modeling*, 7:1393–1512, 1986.

[13] James M. Robins. Non-response models for the analysis of non-monotone non-ignorable missing data. *Statistics in Medicine*, 16:21–37, 1997.

[14] James M. Robins and Mark van der Laan. *Unified Methods for Censored Longitudinal Data and Causality*. Springer-Verlag New York, Inc., 2003.

[15] D. B. Rubin. Causal inference and missing data (with discussion). *Biometrika*, 63:581–592, 1976.

[16] Mauricio Sadinle and Jerome P. Reiter. Itemwise conditionally independent nonresponse modeling for incomplete multivariate data. `https://arxiv.org/abs/1609.00656`, 2016. Working paper.

[17] Ilya Shpitser, Karthika Mohan, and Judea Pearl. Missing data as a causal and probabilistic problem. In *Proceedings of the Thirty First Conference on Uncertainty in Artificial Intelligence (UAI-15)*, pages 802–811. AUAI Press, 2015.

[18] Eric J. Tchetgen Tchetgen, Linbo Wang, and BaoLuo Sun. Discrete choice models for nonmonotone nonignorable missing data: Identification and inference. `https://arxiv.org/abs/1607.02631`, 2016. Working paper.

